# Synchronization and Grammatical Inference in an Oscillating Elman Net

**Bill Baird**
Dept Mathematics,
U.C.Berkeley,
Berkeley, Ca. 94720,
baird@math.berkeley.edu

**Todd Troyer**
Dept Mathematics,
U.C.Berkeley,
Berkeley, Ca. 94720

**Frank Eeckman**
Lawrence Livermore
National Laboratory,
P.O. Box 808 (L-426),
Livermore, Ca. 94551

## Abstract

We have designed an architecture to span the gap between bio-physics and cognitive science to address and explore issues of how a discrete symbol processing system can arise from the continuum, and how complex dynamics like oscillation and synchronization can then be employed in its operation and affect its learning. We show how a discrete-time recurrent "Elman" network architecture can be constructed from recurrently connected oscillatory associative memory modules described by continuous nonlinear ordinary differential equations. The modules can learn connection weights between themselves which will cause the system to evolve under a clocked "machine cycle" by a sequence of transitions of attractors within the modules, much as a digital computer evolves by transitions of its binary flip-flop attractors. The architecture thus employs the principle of "computing with attractors" used by macroscopic systems for reliable computation in the presence of noise. We have specifically constructed a system which functions as a finite state automaton that recognizes or generates the infinite set of six symbol strings that are defined by a Reber grammar. It is a symbol processing system, but with analog input and oscillatory subsymbolic representations. The time steps (machine cycles) of the system are implemented by rhythmic variation (clocking) of a bifurcation parameter. This holds input and "context" modules clamped at their attractors while 'hidden and output modules change state, then clamps hidden and output states while context modules are released to load those states as the new context for the next cycle of input. Superior noise immunity has been demonstrated for systems with dynamic attractors over systems with static attractors, and synchronization ("binding") between coupled oscillatory attractors in different modules has been shown to be important for effecting reliable transitions.

# 1   Introduction

Patterns of 40 to 80 Hz oscillation have been observed in the large scale activity (local field potentials) of olfactory cortex [Freeman and Baird, 1987] and visual neocortex [Gray and Singer, 1987], and shown to predict the olfactory [Freeman and Baird, 1987] and visual pattern recognition responses of a trained animal. Similar observations of 40 Hz oscillation in auditory and motor cortex (in primates), and in the retina and EMG have been reported. It thus appears that cortical computation in general may occur by dynamical interaction of resonant modes, as has been thought to be the case in the olfactory system.

The oscillation can serve a macroscopic clocking function and entrain or "bind" the relevant microscopic activity of disparate cortical regions into a well defined phase coherent collective state or "gestalt". This can overide irrelevant microscopic activity and produce coordinated motor output. There is further evidence that although the oscillatory activity appears to be roughly periodic, it is actually chaotic when examined in detail.

If this view is correct, then oscillatory/chaotic network modules form the actual cortical substrate of the diverse sensory, motor, and cognitive operations now studied in static networks. It must then be shown how those functions can be accomplished with oscillatory and chaotic dynamics, and what advantages are gained thereby. It is our expectation that nature makes good use of this dynamical complexity, and our intent is to search here for novel design principles that may underly the superior computational performance of biological systems over man made devices in many task domains. These principles may then be applied in artificial systems to engineering problems to advance the art of computation. We have therefore constructed a parallel distributed processing architecture that is inspired by the structure and dynamics of cerebral cortex, and applied it to the problem of grammatical inference.

The construction assumes that cortex is a set of coupled oscillatory associative memories, and is also guided by the principle that attractors must be used by macroscopic systems for reliable computation in the presence of noise. Present day digital computers are built of flip-flops which, at the level of their transistors, are continuous dissipative dynamical systems with different attractors underlying the symbols we call "0" and "1".

# 2   Oscillatory Network Modules

The network modules of this architecture were developed previously as models of olfactory cortex, or caricatures of "patches"of neocortex [Baird, 1990a]. A particular subnetwork is formed by a set of neural populations whose interconnections also contain higher order synapses. These synapses determine attractors for that subnetwork independent of other subnetworks. Each subnetwork module assumes only minimal coupling justified by known olfactory anatomy. An $N$ node module can be shown to function as an associative memory for up to $N/2$ oscillatory and $N/3$ chaotic memory attractors [Baird, 1990b, Baird and Eeckman, 1992b]. Single modules with static, oscillatory, and three types of chaotic attractors – Lorenz, Roessler, Ruelle-Takens – have been sucessfully used for recognition of handwritten characters [Baird and Eeckman, 1992b].

We have shown in these modules a superior stability of oscillatory attractors over static attractors in the presence of additive Gaussian noise perturbations with the 1/f spectral character of the noise found experimentally by Freeman in the brain[Baird and Eeckman, 1992a]. This may be one reason why the brain uses dynamic attractors. An oscillatory attractor acts like a a bandpass filter and is

effectively immune to the many slower macroscopic bias perturbations in the theta-alpha-beta range (3 - 25 Hz) below its 40 -80 Hz passband, and the more microscopic perturbations of single neuron spikes in the 100 - 1000 Hz range.

The mathematical foundation for the construction of network modules is contained in the **normal form projection algorithm**[Baird, 1990b]. This is a learning algorithm for recurrent analog neural networks which allows associative memory storage of analog patterns, continuous periodic sequences, and chaotic attractors in the same network. A key feature of a net constructed by this algorithm is that the underlying dynamics is explicitly isomorphic to any of a class of standard, well understood nonlinear dynamical systems - a "normal form" [Guckenheimer and Holmes, 1983]. This system is chosen in advance, independent of both the patterns to be stored and the learning algorithm to be used. This control over the dynamics permits the design of important aspects of the network dynamics independent of the particular patterns to be stored. Stability, basin geometry, and rates of convergence to attractors can be programmed in the standard dynamical system.

By analyzing the network in the polar form of these "normal form coordinates", the amplitude and phase dynamics have a particularly simple interaction. When the input to a module is synchronized with its intrinsic oscillation, the amplitudes of the periodic activity may be considered separately from the phase rotation, and the network of the module may be viewed as a static network with these amplitudes as its activity. We can further show analytically that the network modules we have constructed have a strong tendency to synchronize as required.

## 3   Oscillatory Elman Architecture

Because we work with this class of mathematically well-understood associative memory networks, we can take a constructive approach to building a cortical computer architecture, using these networks as modules in the same way that digital computers are designed from well behaved continuous analog flip-flop circuits. The architecture is such that the larger system is itself a special case of the type of network of the submodules, and can be analysed with the same tools used to design the subnetwork modules.

Each module is described in normal form or "mode" coordinates as a k-winner-take-all network where the winning set of units may have static, periodic or chaotic dynamics. By choosing modules to have only two attractors, networks can be built which are similar to networks using binary units. There can be fully recurrent connections between modules. The entire super-network of connected modules, however, is itself a polynomial network that can be projected into standard network coordinates. The attractors within the modules may then be distributed patterns like those described for the biological model [Baird, 1990a], and observed experimentally in the olfactory system [Freeman and Baird, 1987]. The system is still equivalent to the architecture of modules in normal form, however, and may easily be designed, simulated, and theoretically evaluated in these coordinates. *In this paper all networks are discussed in normal form coordinates.*

As a benchmark for the capabilities of the system, and to create a point of contact to standard network architectures, we have constructed a discrete-time recurrent "Elman" network [Elman, 1991] from oscillatory modules defined by ordinary differential equations. We have at present a system which functions as a finite state automaton that perfectly recognizes or generates the infinite set of strings defined by the Reber grammar described in Cleeremans et. al. [Cleeremans et al., 1989]. The connections for this network were found by psuedo-inverting to find the connection matrices between a set of pre-chosen automata states for the hidden layer modules

and the proper possible output symbols of the Reber grammar, and between the proper next hidden state and each legal combination of a new input symbol and the present state contained in the context modules.

We use two types of modules in implementing the Elman network architecture. The input and output layer each consist of a single associative memory module with six oscillatory attractors (six competing oscillatory modes), one for each of the six possible symbols in the grammar. An attractor in these winner-take-all normal form cordinates is one oscillator at its maximum amplitude, with the others near zero amplitude. The hidden and context layers consist of binary "units" composed of a two competing oscillator module. We think of one mode within the unit as representing "1" and the other as representing "0" (see fig.1).

A "weight" for this unit is simply defined to be the weight of a driving unit to the input of the 1 attractor. The weights for the 0 side of the unit are then given as the compliment of these, $w^0 = A - w^1$. This forces the input to the 0 side of the unit be the complement of the input to the 1 side, $I_i^0 = A - I_i^0$, where A is a bias constant chosen to divide input equally between the oscillators at the midpoint of activation.

**Figure 1.**

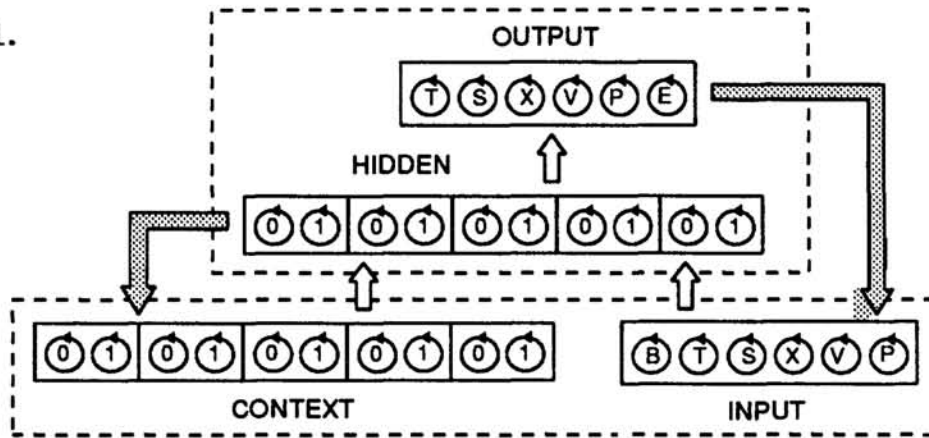

Information flow in the network is controlled by a "machine cycle" implemented by the sinusoidal clocking of a bifurcation parameter which controls the level of inhibitory inter-mode coupling or "competition" between the individual oscillatory modes within each winner-take-all module.

For illustration, we use a binary module represnting either a single hidden or context unit; the behavior of the larger input and output modules is similar. Such a unit is defined in polar normal form coordinates by the following equations:

$$\dot{r}_{1i} = u_i r_{1i} - c r_{1i}(r_{1i}^2 + (d - b\sin(\omega_{clock}t))r_{0i}^2) + \sum_j w_{ij} I_j \cos(\theta_j - \theta_{1i})$$

$$\dot{r}_{0i} = u_i r_{0i} - c r_{0i}(r_{0i}^2 + (d - b\sin(\omega_{clock}t))r_{1i}^2) + \sum_j (A - w_{ij}) I_j \cos(\theta_j - \theta_{0i})$$

$$\dot{\theta}_{1i} = \omega_i + \sum_j w_{ij}(I_j/r_{1i})\sin(\theta_j - \theta_{1i})$$

$$\dot{\theta}_{0i} = \omega_i + \sum_j (A - w_{ij})(I_j/r_{0i})\sin(\theta_j - \theta_{0i})$$

The clocked parameter $bsin(\omega_{clock}t)$ has lower (1/10) frequency than the intrinsic frequency of the unit $\omega_i$. Asuming that all inputs to the unit are phase-locked, examination of the phase equations shows that the unit will synchronize with this input. When the oscillators are phase-locked to the input, $\theta_j - \theta_{1i} = 0$, and the phase terms $\cos(\theta_j - \theta_{1i}) = \cos(0) = 1$ dissappear. This leaves the amplitude equations $\dot{r}_{1i}$ and $\dot{r}_{0i}$ with static inputs $\sum_j w_{ij}I_j$ and $\sum_j(A - w_{ij})I_j$. The phase equations show a strong tendency to phase-lock, since there is an attractor at zero phase difference $\phi = \theta_0 - \theta_I = \theta_0 - \omega_I t = 0$, and a repellor at 180 degrees in the phase difference equations $\dot{\phi}$ for either side of a unit driven by an input of the same frequency, $\omega_I - \omega_0 = 0$.

$$\dot{\phi} = \omega_0 - \omega_I + (r_I/r_0)\sin(-\phi) \;, \quad so\;, \quad \hat{\phi} = -sin^{-1}[(r_0/r_I)(\omega_I - \omega_0)]$$

Thus we have a network module which approximates a static network unit in its amplitude activity when fully phase-locked. Amplitude information is transmitted between modules, with an oscillatory carrier. If the frequencies of attractors in the architecture are randomly dispersed by a significant amount, phase-lags appear, then synchronization is lost and improper transitions begin to occur.

*For the remainder of the paper we assume the entire system is operating in the synchronized regime and examine the flow of information characterized by the pattern of amplitudes of the oscillatory modes within the network.*

## 4    Machine Cycle by Clocked Bifurcation

Given this assumption of a phase-locked system, the amplitude dynamics behave as a gradient dynamical system for an energy function given by

$$E_i = -\frac{1}{2}u_i(r_{1i}^2 + r_{0i}^2) + \frac{1}{4}c((r_{1i}^4 + r_{0i}^4) + \frac{1}{2}(d \pm bsin(\omega_{clock}t))(r_{1i}^2 r_{0i}^2) - (Ir_{1i} + (B-I)r_{0i})$$

where the total input $I = \sum_j w_{ij}I_j$ and $B = \sum_j I_j$. Figures 2a and 2b show the energy landscape with no external input for minimal and maximal levels of competition respectively. External input simply adds a linear "tilt" to the landscape, with large $I$ giving a larger tilt toward the $r_{1i}$ axis and small $I$ a larger tilt toward the $r_{0i}$ axis.

Note that for low levels of competition, there is a broad circular valley. When tilted by external input, there is a unique equilibrium that is determined by the bias in tilt along one axis over the other. Thinking of $r_{1i}$ as the "acitivity" of the unit, this acitivity becomes an increasing function of $I$. *The module behaves as analog connectionist unit whose transfer function can be approximated by a sigmoid.*

With high levels of competition, the unit will behave as a binary (bistable) "digital" flip-flop element. There are two deep valleys, one on each axis. Hence the final steady state of the unit is determined by which basin contains the initial state of the system reached during the analog mode of operation before competition is increased by the clock. This state changes little under the influence of external input: a tilt will move the location of the valleys only slightly. Hence *the unit performs a winner-take-all choice on the coordinates of its initial state and maintains that choice independent of external input.*

**Figure 2a.**

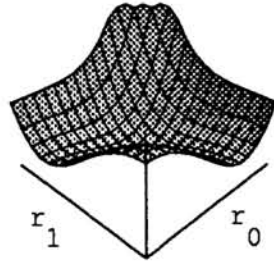

Low
Competition        r₁        r₀

**Figure 2b.**

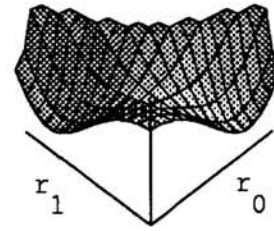

High
Competition        r₁        r₀

We use this bifurcation in the behavior of the modules to control information flow within the network. We think of the input and context modules as "sensory", and the hidden and output modules as "motor" modules. The action of the clock is applied reciprocally to these two sets (grouped by dotted lines in fig.1) so that they alternatively open to receive input from each other and make transitions of attractors. *This enables a network completely defined as a set of ordinary differential equations to implement the discrete-time recurrent Elman network.*

At the beginning of a machine cycle, the input and context layers are at high competition and hence their activity is "clamped" at the bottom of deep attractors. The hidden and output modules are at low competition and therefore behave as traditional feedforward network free to take on analog values. Then the situation reverses. As the competition comes up in the output module, it makes a winner-take-all choice as to the next symbol. Meanwhile high competition has quantized and clamped the activity in the hidden layer to a fixed binary vector. Then competition is lowered in the input and context layers, freeing these modules from their attractors.

Identity mappings from hidden to context and from output to input (gray arrows in fig.1) "load" the binarized activity of the hidden layer to the context layer for the next cycle, and "place" the generated output symbol into the input layer. For a Reber grammar there are always two equally possible next symbols being generated in the output layer, and we apply noise to break this symmetry and let the winner-take-all dynamics of the output module chose one. For the recognition mode of operation, these symbols are thought of as "predicted" by the output, and one of them must always match the next actual input of a string to be recognized or the string is instantly rejected.

Note that even though the clocking is sinusiodal and these transitions are not sharp, the system is robust and reliable. It is only necessary to set the rates of convergence within modules to be faster than the rate of change of the clocked bifurcation parameter, so that the modules are operating "adiabatically" – i.e. always internally relaxed to an equilibrium that is moved slowly by the clocked parameter.

It is the bifurcation in the phase portrait of a module from one to two attractors that contributes the essential "digitization" of the system in time and state. A bifurcation is a discontinuous (topologically sharp) change in the phase portrait of possibilities for the continuous dynamical behavior of a system that occurs as a bifurcation parameter reaches a "critical" value. We can think of the analog mode for a module as allowing input to prepare its initial state for the binary "decision" between attractor basins that occurs as competition rises and the double potential well appears.

The feedback between sensory and motor modules is effectively cut when one set is clamped at high competition. The system can thus be viewed as operating in discrete time by alternating transitions between a finite set of attracting states. This kind of clocking and "buffering" (clamping) of some states while other states

relax is essential to the reliable operation of digital architectures. The clock input on a flip-flop clamps it's state until its signal inputs have settled and the choice of transition can be made with the proper information available. In our simulations, if we clock all modules to transition at once, the programmed sequences lose stability, and we get transitions to unprogrammed fixed points and simple limit cycles for the whole system.

## 5    Training

When the input and context modules are clamped at their attractors, and the hidden and output modules are in the analog operating mode and synchronized to their inputs, the network approximates the behavior of a standard feedforward network in terms of its amplitude activities. Thus a real valued error can be defined for the hidden and output units and standard learning algorithms like backpropagation can be used to train the connections.

We can use techniques of Giles et. al. [Giles et al., 1992] who have trained simple recurrent networks to become finite state automata that can recognize the regular Tomita languages and others. If the context units are clamped with high competition, they are essentially "quantized" to take on only their 0 or 1 attractor values, and the feedback connections from the hidden units cannot affect them. While Giles, et. al. often do not quantize their units until the end of training to extract a finite state automaton, they find that quantizing of the context units during training like this increases learning speed in many cases[Giles et al., 1992]. In preparation for learning in the dynamic architecture, we have sucessfully trained the backpropogation network of Cleermans et. al. with digititized context units and a shifted sigmoid activation function that approximates the one calculated for our oscillatory units.

In the dynamic architecture, we have also the option of leaving the competition within the context units at intermediate levels to allow them to take on analog values in a variable sized neighborhood of the 0 or 1 attractors. Since our system is recurrently connected by an identity map from hidden to context units, it will relax to some equilibrium determined by the impact of the context units and the clamped input on the hidden unit states, and the effect of the feedback from those hidden states on the context states. We can thus further explore the impact on learning of this range of operation between discrete time and space automaton and continuous analog recurrent network.

## 6    Discusion

The ability to operate as an finite automaton with oscillatory/chaotic "states" is an important benchmark for this architecture, but only a subset of its capabilities. At low to zero competition, the supra-system reverts to one large continuous dynamical system. We expect that this kind of variation of the operational regime, especially with chaotic attractors inside the modules, though unreliable for habitual behaviors, may nontheless be very useful in other areas such as the search process of reinforcement learning.

An important element of intra-cortical communication in the brain, and between modules in this architecture, is the ability of a module to detect and respond to the proper input signal from a particular module, when inputs from other modules which is irrelevant to the present computation are contributing cross-talk and noise. This is smilar to the problem of coding messages in a computer architecture like the

connection machine so that they can be picked up from the common communication buss line by the proper receiving module. We believe that sychronization is one important aspect of how the brain solves this coding problem. Attractors in modules of the architecture may be frequency coded during learning so that they will sychronize only with the appropriate active attractors in other modules that have a similar resonant frequency. The same hardware (or "wetware") and connection matrix can thus subserve many different networks of interaction between modules at the same time without cross-talk problems.

This type of computing architecture and its learning algorithms for computation with oscillatory spatial modes may be ideal for implementation in optical systems, where electromagnetic oscillations, very high dimensional modes, and high processing speeds are available. The mathematical expressions for optical mode competition are nearly identical to our normal forms.

## Acknowledgements

Supported by AFOSR-91-0325, and a grant from LLNL. It is a pleasure to acknowledge the invaluable assistance of Morris Hirsch and Walter Freeman.

## References

[Baird, 1990a] Baird, B. (1990a). Bifurcation and learning in network models of oscillating cortex. In Forest, S., editor, *Emergent Computation*, pages 365–384. North Holland. also in Physica D, 42.

[Baird, 1990b] Baird, B. (1990b). A learning rule for cam storage of continuous periodic sequences. In *Proc. Int. Joint Conf. on Neural Networks, San Diego*, pages 3: 493–498.

[Baird and Eeckman, 1992a] Baird, B. and Eeckman, F. H. (1992a). A hierarchical sensory-motor architecture of oscillating cortical area subnetworks. In Eeckman, F. H., editor, *Analysis and Modeling of Neural Systems II*, pages 96–204, Norwell, Ma. Kluwer.

[Baird and Eeckman, 1992b] Baird, B. and Eeckman, F. H. (1992b). A normal form projection algorithm for associative memory. In Hassoun, M. H., editor, *Associative Neural Memories: Theory and Implementation*, New York, NY. Oxford University Press. in press.

[Cleeremans et al., 1989] Cleeremans, A., Servan-Schreiber, D., and McClelland, J. (1989). Finite state automata and simple recurrent networks. *Neural Computation*, 1(3):372–381.

[Elman, 1991] Elman, J. (1991). Distributed representations, simple recurrent networks and grammatical structure. *Machine Learning*, 7(2/3):91.

[Freeman and Baird, 1987] Freeman, W. and Baird, B. (1987). Relation of olfactory eeg to behavior: Spatial analysis. *Behavioral Neuroscience*, 101:393–408.

[Giles et al., 1992] Giles, C., Miller, C.B.and Chen, D., Chen, H., Sun, G., and Lee, Y. (1992). Learning and extracting finite state automata with second order recurrent neural networks. *Neural Computation*, pages 393–405.

[Gray and Singer, 1987] Gray, C. M. and Singer, W. (1987). Stimulus dependent neuronal oscillations in the cat visual cortex area 17. *Neuroscience [Suppl]*, 22:1301P.

[Guckenheimer and Holmes, 1983] Guckenheimer, J. and Holmes, D. (1983). *Nonlinear Oscillations, Dynamical Systems, and Bifurcations of Vector Fields*. Springer, New York.